# Learning from Infinite Data in Finite Time

**Pedro Domingos**  **Geoff Hulten**
Department of Computer Science and Engineering
University of Washington
Seattle, WA 98185-2350, U.S.A.
{*pedrod, ghulten*}*@cs.washington.edu*

## Abstract

We propose the following general method for scaling learning algorithms to arbitrarily large data sets. Consider the model $M_{\vec{n}}$ learned by the algorithm using $n_i$ examples in step $i$ ($\vec{n} = (n_1, \ldots, n_m)$), and the model $M_{\infty}$ that would be learned using infinite examples. Upper-bound the loss $L(M_{\vec{n}}, M_{\infty})$ between them as a function of $\vec{n}$, and then minimize the algorithm's time complexity $f(\vec{n})$ subject to the constraint that $L(M_{\infty}, M_{\vec{n}})$ be at most $\epsilon$ with probability at most $\delta$. We apply this method to the EM algorithm for mixtures of Gaussians. Preliminary experiments on a series of large data sets provide evidence of the potential of this approach.

## 1 An Approach to Large-Scale Learning

Large data sets make it possible to reliably learn complex models. On the other hand, they require large computational resources to learn from. While in the past the factor limiting the quality of learnable models was typically the quantity of data available, in many domains today data is super-abundant, and the bottleneck is the time required to process it. Many algorithms for learning on large data sets have been proposed, but in order to achieve scalability they generally compromise the quality of the results to an unspecified degree. We believe this unsatisfactory state of affairs is avoidable, and in this paper we propose a general method for scaling learning algorithms to arbitrarily large databases without compromising the quality of the results. Our method makes it possible to learn in finite time a model that is essentially indistinguishable from the one that would be obtained using infinite data.

Consider the simplest possible learning problem: estimating the mean of a random variable $x$. If we have a very large number of samples, most of them are probably superfluous. If we are willing to accept an error of at most $\epsilon$ with probability at most $\delta$, Hoeffding bounds [4] (for example) tell us that, irrespective of the distribution of $x$, only $n = \frac{1}{2}(R/\epsilon)^2 \ln(2/\delta)$ samples are needed, where $R$ is $x$'s range. We propose to extend this type of reasoning beyond learning single parameters, to learning complex models. The approach we propose consists of three steps:

1. Derive an upper bound on the relative loss between the finite-data and infinite-data models, as a function of the number of samples used in each step of the finite-data algorithm.

2. Derive an upper bound on the time complexity of the learning algorithm, as a function of the number of samples used in each step.

3. Minimize the time bound (via the number of samples used in each step) subject to target limits on the loss.

In this paper we exemplify this approach using the EM algorithm for mixtures of Gaussians. In earlier papers we applied it (or an earlier version of it) to decision tree induction [2] and $k$-means clustering [3]. Despite its wide use, EM has long been criticized for its inefficiency (see discussion following Dempster et al. [1]), and has been considered unsuitable for large data sets [8]. Many approaches to speeding it up have been proposed (see Thiesson et al. [6] for a survey). Our method can be seen as an extension of *progressive sampling* approaches like Meek et al. [5]: rather than minimize the total number of samples needed by the algorithm, we minimize the number needed by each step, leading to potentially much greater savings; and we obtain guarantees that do not depend on unverifiable extrapolations of learning curves.

## 2    A Loss Bound for EM

In a mixture of Gaussians model, each $D$-dimensional data point $x_j$ is assumed to have been independently generated by the following process: 1) randomly choose a mixture component $k$; 2) randomly generate a point from it according to a Gaussian distribution with mean $\mu_k$ and covariance matrix $\Sigma_k$. In this paper we will restrict ourselves to the case where the number $K$ of mixture components and the probability of selection $p(\mu_k)$ and covariance matrix for each component are known. Given a training set $S = \{x_1, \ldots, x_N\}$, the learning goal is then to find the maximum-likelihood estimates of the means $\mu_k$. The EM algorithm [1] accomplishes this by, starting from some set of initial means, alternating until convergence between estimating the probability $p(\mu_k|x_j)$ that each point was generated by each Gaussian (the E step), and computing the ML estimates of the means $\hat{\mu}_k = \sum_{j=1}^{N} w_{jk} x_j / \sum_{j=1}^{N} w_{jk}$ (the M step), where $w_{jk} = p(\mu_k|x_j)$ from the previous E step. In the basic EM algorithm, all $N$ examples in the training set are used in each iteration. The goal in this paper is to speed up EM by using only $n_i < N$ examples in the $i$th iteration, while guaranteeing that the means produced by the algorithm do not differ significantly from those that would be obtained with arbitrarily large $N$.

Let $M_{\vec{n}} = (\hat{\mu}_1, \ldots, \hat{\mu}_K)$ be the vector of mean estimates obtained by the finite-data EM algorithm (i.e., using $n_i$ examples in iteration $i$), and let $M_\infty = (\mu_1, \ldots, \mu_K)$ be the vector obtained using infinite examples at each iteration. In order to proceed, we need to quantify the difference between $M_{\vec{n}}$ and $M_\infty$. A natural choice is the sum of the squared errors between corresponding means, which is proportional to the negative log-likelihood of the finite-data means given the infinite-data ones:

$$L(M_{\vec{n}}, M_\infty) = \sum_{k=1}^{K} ||\hat{\mu}_k - \mu_k||^2 = \sum_{k=1}^{K} \sum_{d=1}^{D} |\hat{\mu}_{kd} - \mu_{kd}|^2 \qquad (1)$$

where $\hat{\mu}_{kd}$ is the $d$th coordinate of $\hat{\mu}$, and similarly for $\mu_{kd}$.

After any given iteration of EM, $|\hat{\mu}_{kd} - \mu_{kd}|$ has two components. One, which we call the *sampling error*, derives from the fact that $\hat{\mu}_{kd}$ is estimated from a finite sample,

while $\mu_{kd}$ is estimated from an infinite one. The other component, which we call the *weighting error*, derives from the fact that, due to sampling errors in previous iterations, the weights $w_{jk}$ used to compute the two estimates may differ. Let $\mu_{kdi}$ be the infinite-data estimate of the $d$th coordinate of the $k$th mean produced in iteration $i$, $\hat{\mu}_{kdi}$ be the corresponding finite-data estimate, and $\bar{\mu}_{kdi}$ be the estimate that would be obtained if there were no weighting errors in that iteration. Then the sampling error at iteration $i$ is $|\bar{\mu}_{kdi} - \mu_{kdi}|$, the weighting error is $|\hat{\mu}_{kdi} - \bar{\mu}_{kdi}|$, and the total error is $|\hat{\mu}_{kdi} - \mu_{kdi}| \leq |\hat{\mu}_{kdi} - \bar{\mu}_{kdi}| + |\bar{\mu}_{kdi} - \mu_{kdi}|$.

Given bounds on the total error of each coordinate of each mean after iteration $i-1$, we can derive a bound on the weighting error after iteration $i$ as follows. Bounds on $\mu_{kd,i-1}$ for each $d$ imply bounds on $p(x_j|\mu_{ki})$ for each example $x_j$, obtained by substituting the maximum and minimum allowed distances between $x_{jd}$ and $\mu_{kd,i-1}$ into the expression of the Gaussian distribution. Let $p_{jki}^+$ be the upper bound on $p(x_j|\mu_{ki})$, and $p_{jki}^-$ be the lower bound. Then the weight of example $x_j$ in mean $\mu_{ki}$ can be bounded from below by $w_{jki}^- = p_{jki}^- p(\mu_k) / \sum_{k'=1}^{K} p_{jk'i}^+ p(\mu_k')$, and from above by $w_{jki}^+ = \min\{p_{jki}^+ p(\mu_k) / \sum_{k'=1}^{K} p_{jk'i}^- p(\mu_k'), 1\}$. Let $w_{jki}^{(+)} = w_{jki}^+$ if $x_j \geq 0$ and $w_{jki}^{(+)} = w_{jki}^-$ otherwise, and let $w_{jki}^{(-)} = w_{jki}^-$ if $x_j \geq 0$ and $w_{jki}^{(-)} = w_{jki}^+$ otherwise. Then

$$
\begin{aligned}
|\hat{\mu}_{kdi} - \bar{\mu}_{kdi}| &= \left| \hat{\mu}_{kdi} - \frac{\sum_{j=1}^{n_i} w_{jki} x_j}{\sum_{j=1}^{n_i} w_{jki}} \right| \\
&\leq \max\left\{ \left| \hat{\mu}_{kdi} - \frac{\sum_{j=1}^{n_i} w_{jki}^{(+)} x_j}{\sum_{j=1}^{n_i} w_{jki}^-} \right|, \left| \hat{\mu}_{kdi} - \frac{\sum_{j=1}^{n_i} w_{jki}^{(-)} x_j}{\sum_{j=1}^{n_i} w_{jki}^+} \right| \right\}
\end{aligned}
\quad (2)
$$

A corollary of Hoeffding's [4] Theorem 2 is that, with probability at least $1 - \delta$, the sampling error is bounded by

$$
|\bar{\mu}_{kdi} - \mu_{kdi}| \leq \sqrt{\frac{R_d^2 \ln(2/\delta) \sum_{j=1}^{n_i} w_{jki}^2}{2(\sum_{j=1}^{n_i} w_{jki})^2}} \leq \sqrt{\frac{R_d^2 \ln(2/\delta) \sum_{j=1}^{n_i} (w_{jki}^+)^2}{2(\sum_{j=1}^{n_i} w_{jki}^-)^2}}
\quad (3)
$$

where $R_d$ is the range of the $d$th coordinate of the data (assumed known[1]). This bound is independent of the distribution of the data, which will ensure that our results are valid even if the data was not truly generated by a mixture of Gaussians, as is often the case in practice. On the other hand, the bound is more conservative than distribution-dependent ones, requiring more samples to reach the same guarantees.

The initialization step is error-free, assuming the finite- and infinite-data algorithms are initialized with the same means. Therefore the weighting error in the first iteration is zero, and Equation 3 bounds the total error. From this we can bound the weighting error in the second iteration according to Equation 2, and therefore bound the total error by the sum of Equations 2 and 3, and so on for each iteration until the algorithms converge. If the finite- and infinite-data EM converge in the same number of iterations $m$, the loss due to finite data is $L(M_{\bar{n}}, M_\infty) = \sum_{k=1}^{K} \sum_{d=1}^{D} |\hat{\mu}_{kdm} - \mu_{kdm}|^2$ (see Equation 1). Assume that the convergence criterion is $\sum_{k=1}^{K} \|\mu_{ki} - \mu_{k,i-1}\|^2 \leq \gamma$. In general

(with probability specified below), infinite-data EM converges at one of the iterations for which the minimum possible change in mean positions is below $\gamma$, and is guaranteed to converge at the first iteration for which the maximum possible change is below $\gamma$. More precisely, it converges at one of the iterations for which $\sum_{k=1}^{K} \sum_{d=1}^{D} (\max\{|\hat{\mu}_{kd,i-1} - \hat{\mu}_{kdi}| - |\hat{\mu}_{kd,i-1} - \mu_{kd,i-1}| - |\hat{\mu}_{kdi} - \mu_{kdi}|, 0\})^2 \leq \gamma$, and is guaranteed to converge at the first iteration for which $\sum_{k=1}^{K} \sum_{d=1}^{D} (|\hat{\mu}_{kd,i-1} - \hat{\mu}_{kdi}| + |\hat{\mu}_{kd,i-1} - \mu_{kd,i-1}| + |\hat{\mu}_{kdi} - \mu_{kdi}|)^2 \leq \gamma$. To obtain a bound for $L(M_{\vec{n}}, M_{\infty})$, finite-data EM must be run until the latter condition holds. Let $I$ be the set of iterations at which infinite-data EM could have converged. Then we finally obtain

$$L(M_{\vec{n}}, M_{\infty}) \leq \max_{i \in I} \left\{ \sum_{k=1}^{K} \sum_{d=1}^{D} (|\hat{\mu}_{kdi} - \hat{\mu}_{kdm}| + |\hat{\mu}_{kdi} - \mu_{kdi}|)^2 \right\} \qquad (4)$$

where $m$ is the total number of iterations carried out. This bound holds if all of the Hoeffding bounds (Equation 3) hold. Since each of these bounds fails with probability at most $\delta$, the bound above fails with probability at most $\delta^* = KDm\delta$ (by the union bound). As a result, the growth with $K$, $D$ and $m$ of the number of examples required to reach a given loss bound with a given probability is only $O(\sqrt{\ln KDm})$.

The bound we have just derived utilizes run-time information, namely the distance of each example to each mean along each coordinate in each iteration. This allows it to be tighter than *a priori* bounds. Notice also that it would be trivial to modify the treatment for any other loss criterion that depends only on the terms $|\hat{\mu}_{kdm} - \mu_{kdm}|$ (e.g., absolute loss).

## 3   A Fast EM Algorithm

We now apply the previous section's result to reduce the number of examples used by EM at each iteration while keeping the loss bounded. We call the resulting algorithm VFEM. The goal is to learn in minimum time a model whose loss relative to EM applied to infinite data is at most $\epsilon^*$ with probability at least $1 - \delta^*$. (The reason to use $\epsilon^*$ instead of $\epsilon$ will become apparent below.) Using the notation of the previous section, if $n_i$ examples are used at each iteration then the running time of EM is $O(KD \sum_{i=1}^{m} n_i)$, and can be minimized by minimizing $\sum_{i=1}^{m} n_i$. Assume for the moment that the number of iterations $m$ is known. Then, using Equation 1, we can state the goal more precisely as follows.

**Goal:** *Minimize $\sum_{i=1}^{m} n_i$, subject to the constraint that $\sum_{k=1}^{K} ||\hat{\mu}_{km} - \mu_{km}||^2 \leq \epsilon^*$ with probability at least $1 - \delta^*$.*

A sufficient condition for $\sum_{k=1}^{K} ||\hat{\mu}_{km} - \mu_{km}||^2 \leq \epsilon^*$ is that $\forall k \;\; ||\hat{\mu}_{km} - \mu_{km}|| \leq \sqrt{\epsilon^*/K}$. We thus proceed by first minimizing $\sum_{i=1}^{m} n_i$ subject to $||\hat{\mu}_{km} - \mu_{km}|| \leq \sqrt{\epsilon^*/K}$ separately for each mean.[2] In order to do this, we need to express $||\hat{\mu}_{km} - \mu_{km}||$ as a function of the $n_i$'s. By the triangle inequality, $||\hat{\mu}_{ki} - \mu_{ki}|| \leq ||\hat{\mu}_{ki} - \bar{\mu}_{ki}|| + ||\bar{\mu}_{ki} - \mu_{ki}||$. By Equation 3, $||\bar{\mu}_{ki} - \mu_{ki}|| \leq \sqrt{\frac{1}{2} R^2 \ln(2/\delta) \sum_{j=1}^{n_i} w_{jki}^2 / (\sum_{j=1}^{n_i} w_{jki})^2}$, where $R^2 = \sum_{d=1}^{D} R_d^2$ and $\delta = \delta^*/KDm$ per the discussion following Equation 4. The $(\sum_{j=1}^{n_i} w_{jki})^2 / \sum_{j=1}^{n_i} w_{jki}^2$ term is a measure of the diversity of the weights,

being equal to $1 - 1/\text{Gini}(W'_{ki})$, where $W'_{ki}$ is the vector of normalized weights $w'_{jki} = w_{jki}/\sum_{j'=1}^{n_i} w_{j'ki}$. It attains a minimum of 1 when all the weights but one are zero, and a maximum of $n_i$ when all the weights are equal and non-zero. However, we would like to have a measure whose maximum is independent of $n_i$, so that it remains approximately constant whatever the value of $n_i$ chosen (for sufficiently large $n_i$). The measure will then depend only on the underlying distribution of the data. Thus we define $\beta_{ki} = (\sum_{j=1}^{n_i} w_{jki})^2/(n_i \sum_{j=1}^{n_i} w_{jki}^2)$, obtaining $\|\bar{\mu}_{ki} - \mu_{ki}\| \leq \sqrt{R^2 \ln(2/\delta)/(2\beta_{ki}n_i)}$. Also, $\|\hat{\mu}_{ki} - \bar{\mu}_{ki}\| = \sqrt{\sum_{d=1}^{D} |\hat{\mu}_{kdi} - \bar{\mu}_{kdi}|^2}$, with $|\hat{\mu}_{kdi} - \bar{\mu}_{kdi}|$ bounded by Equation 2. To keep the analysis tractable, we upper-bound this term by a function proportional to $\|\hat{\mu}_{kd,i-1} - \mu_{kd,i-1}\|$. This captures the notion than the weighting error in one iteration should increase with the total error in the previous one. Combining this with the bound for $\|\bar{\mu}_{ki} - \mu_{ki}\|$, we obtain

$$\|\hat{\mu}_{ki} - \mu_{ki}\| \leq \alpha_{ki} \|\hat{\mu}_{k,i-1} - \mu_{k,i-1}\| + \sqrt{\frac{R^2 \ln(2/\delta)}{2\beta_{ki}n_i}} \tag{5}$$

where $\alpha_{ki}$ is the proportionality constant. Given this equation and $\|\hat{\mu}_{k0} - \mu_{k0}\| = 0$, it can be shown by induction that

$$\|\hat{\mu}_{km} - \mu_{km}\| \leq \sum_{i=1}^{m} \frac{r_{ki}}{\sqrt{n_i}} \tag{6}$$

where

$$r_{ki} = \sqrt{\frac{R^2 \ln(2/\delta)}{2\beta_{ki}}} \prod_{j=i+1}^{m} \alpha_{kj} \tag{7}$$

The target bound will thus be satisfied by minimizing $\sum_{i=1}^{m} n_i$ subject to $\sum_{i=1}^{m} (r_{ki}/\sqrt{n_i}) = \sqrt{\epsilon^*/K}$.[3] Finding the $n_i$'s by the method of Lagrange multipliers yields

$$n_i = \frac{K}{\epsilon^*} \left( \sum_{j=1}^{m} \sqrt[3]{r_{ki}r_{kj}^2} \right)^2 \tag{8}$$

This equation will produce a required value of $n_i$ for each mean. To guarantee the desired $\epsilon^*$, it is sufficient to make $n_i$ equal to the maximum of these values.

The VFEM algorithm consists of a sequence of runs of EM, with each run using more examples than the last, until the bound $L(M_{\vec{n}}, M_\infty) \leq \epsilon^*$ is satisfied, with $L(M_{\vec{n}}, M_\infty)$ bounded according to Equation 4. In the first run, VFEM postulates a maximum number of iterations $m$, and uses it to set $\delta = \delta^*/KDm$. If $m$ is exceeded, for the next run it is set to 50% more than the number needed in the current run. (A new run will be carried out if either the $\delta^*$ or $\epsilon^*$ target is not met.) The number of examples used in the first run of EM is the same for all iterations, and is set to $1.1(K/2)(R/\epsilon^*)^2 \ln(2/\delta)$. This is 10% more than the number of examples that would theoretically be required in the best possible case (no weighting errors in the last

iteration, leading to a pure Hoeffding bound, and a uniform distribution of examples among mixture components). The numbers of examples for subsequent runs are set according to Equation 8. For iterations beyond the last one in the previous run, the number of examples is set as for the first run. A run of EM is terminated when $\sum_{k=1}^{K} \sum_{d=1}^{D} (|\hat{\mu}_{kd,i-1} - \hat{\mu}_{kdi}| + |\hat{\mu}_{kd,i-1} - \mu_{kd,i-1}| + |\hat{\mu}_{kdi} - \mu_{kdi}|)^2 \leq \gamma$ (see discussion preceding Equation 4), or two iterations after $\sum_{k=1}^{K} ||\mu_{ki} - \mu_{k,i-1}||^2 \leq \gamma/3$, whichever comes first. The latter condition avoids overly long unproductive runs. If the user target bound is $\epsilon$, $\epsilon^*$ is set to $\min\{\epsilon, \gamma/3\}$, to facilitate meeting the first criterion above. When the convergence threshold for infinite-data EM was not reached even when using the whole training set, VFEM reports that it was unable to find a bound; otherwise the bound obtained is reported.

VFEM ensures that the total number of examples used in one run is always at least twice the number $n$ used in the previous run. This is done by, if $\sum n_i < 2n$, setting the $n_i$'s instead to $n_i' = 2n(n_i / \sum n_i)$. If at any point $\sum n_i > mN$, where $m$ is the number of iterations carried out and $N$ is the size of the full training set, $\forall i \; n_i = N$ is used. Thus, assuming that the number of iterations does not decrease with the number of examples, VFEM's total running time is always less than three times the time taken by the last run of EM. (The worst case occurs when the one-but-last run is carried out on almost the full training set.)

The run-time information gathered in one run is used to set the $n_i$'s for the next run. We compute each $\alpha_{ki}$ as $||\hat{\mu}_{ki} - \bar{\mu}_{ki}|| / ||\hat{\mu}_{k,i-1} - \mu_{k,i-1}||$. The approximations made in the derivation will be good, and the resulting $n_i$'s accurate, if the means' paths in the current run are similar to those in the previous run. This may not be true in the earlier runs, but their running time will be negligible compared to that of later runs, where the assumption of path similarity from one run to the next should hold.

# 4   Experiments

We conducted a series of experiments on large synthetic data sets to compare VFEM with EM. All data sets were generated by mixtures of spherical Gaussians with means $\mu_k$ in the unit hypercube. Each data set was generated according to three parameters: the dimensionality $D$, the number of mixture components $K$, and the standard deviation $\sigma$ of each coordinate in each component. The means were generated one at a time by sampling each dimension uniformly from the range $(2\sigma, 1 - 2\sigma)$. This ensured that most of the data points generated were within the unit hypercube. The range of each dimension in VFEM was set to one. Rather than discard points outside the unit hypercube, we left them in to test VFEM's robustness to outliers. Any $\mu_k$ that was less than $(\sqrt{D}/K)\sigma$ away from a previously generated mean was rejected and regenerated, since problems with very close means are unlikely to be solvable by either EM or VFEM. Examples were generated by choosing one of the means $\mu_k$ with uniform probability, and setting the value of each dimension of the example by randomly sampling from a Gaussian distribution with mean $\mu_{kd}$ and standard deviation $\sigma$. We compared VFEM to EM on 64 data sets of 10 million examples each, generated by using every possible combination of the following parameters: $D \in \{4, 8, 12, 16\}$; $K \in \{3, 4, 5, 6\}$; $\sigma \in \{.01, .03, .05, .07\}$. In each run the two algorithms were initialized with the same means, randomly selected with the constraint that no two be less than $\sqrt{D}/(2K)$ apart. VFEM was allowed to converge before EM's guaranteed convergence criterion was met (see discussion preceding Equation 4). All experiments were run on a 1 GHz Pentium III machine under Linux, with $\gamma = 0.0001DK$, $\delta^* = 0.05$, and $\epsilon^* = \min\{0.01, \gamma\}$.

Table 1: Experimental results. Values are averages over the number of runs shown. Times are in seconds, and #EA is the total number of example accesses made by the algorithm, in millions.

| Runs | Algorithm | #Runs | Time | #EA | Loss | D | K | $\sigma$ |
|------|-----------|-------|------|-----|------|---|---|----------|
| Bound | VFEM | 40 | 217 | 1.21 | 2.51 | 10.5 | 4.2 | 0.029 |
|  | EM | 40 | 3457 | 19.75 | 2.51 | 10.5 | 4.2 | 0.029 |
| No bound | VFEM | 24 | 7820 | 43.19 | 1.20 | 9.1 | 4.9 | 0.058 |
|  | EM | 24 | 4502 | 27.91 | 1.20 | 9.1 | 4.9 | 0.058 |
| All | VFEM | 64 | 3068 | 16.95 | 2.02 | 10 | 4.5 | 0.04 |
|  | EM | 64 | 3849 | 22.81 | 2.02 | 10 | 4.5 | 0.04 |

The results are shown in Table 1. Losses were computed relative to the true means, with the best match between true means and empirical ones found by greedy search. Results for runs in which VFEM achieved and did not achieve the required $\epsilon^*$ and $\delta^*$ bounds are reported separately. VFEM achieved the required bounds and was able to stop early on 62.5% of its runs. When it found a bound, it was on average 16 times faster than EM. When it did not, it was on average 73% slower. The losses of the two algorithms were virtually identical in both situations. VFEM was more likely to converge rapidly for higher $D$'s and lower $K$'s and $\sigma$'s. When achieved, the average loss bound for VFEM was 0.006554, and for EM it was 0.000081. In other words, the means produced by both algorithms were virtually identical to those that would be obtained with infinite data.[4]

We also compared VFEM and EM on a large real-world data set, obtained by recording a week of Web page requests from the entire University of Washington campus. The data is described in detail in Wolman et al. [7], and the preprocessing carried out for these experiments is described in Domingos & Hulten [3]. The goal was to cluster patterns of Web access in order to support distributed caching. On a dataset with $D = 10$ and 20 million examples, with $\delta^* = 0.05$, $\gamma = 0.001$, $\epsilon^* = \gamma/3$, $K = 3$, and $\sigma = 0.01$, VFEM achieved a loss bound of 0.00581 and was two orders of magnitude faster than EM (62 seconds vs. 5928), while learning essentially the same means.

VFEM's speedup relative to EM will generally approach infinity as the data set size approaches infinity. The key question is thus: what are the data set sizes at which VFEM becomes worthwhile? The tentative evidence from these experiments is that they will be in the millions. Databases of this size are now common, and their growth continues unabated, auguring well for the use of VFEM.

## 5   Conclusion

Learning algorithms can be sped up by minimizing the number of examples used in each step, under the constraint that the loss between the resulting model and the one that would be obtained with infinite data remain bounded. In this paper we applied this method to the EM algorithm for mixtures of Gaussians, and observed the resulting speedups on a series of large data sets.

## Acknowledgments

This research was partly supported by NSF CAREER and IBM Faculty awards to the first author, and by a gift from the Ford Motor Company.

## Footnotes

[1]Although a normally distributed variable has infinite range, our experiments show that assuming a sufficiently wide finite range does not significantly affect the results.

[2]This will generally lead to a suboptimal solution; improving it is a matter for future work.

[3]This may lead to a suboptimal solution for the $n_i$'s, in the unlikely case that $\|\hat{\mu}_{km} - \mu_{km}\|$ increases with them.

[4]The much higher loss values relative to the true means, however, indicate that infinite-data EM would often find only local optima (unless the greedy search itself only found a suboptimal match).

## References

[1] A. P. Dempster, N. M. Laird, and D. B. Rubin. Maximum likelihood from incomplete data via the EM algorithm. *Journal of the Royal Statistical Society, Series B*, 39:1–38, 1977.

[2] P. Domingos and G. Hulten. Mining high-speed data streams. In *Proceedings of the Sixth ACM SIGKDD International Conference on Knowledge Discovery and Data Mining*, pp. 71–80, Boston, MA, 2000. ACM Press.

[3] P. Domingos and G. Hulten. A general method for scaling up machine learning algorithms and its application to clustering. In *Proceedings of the Eighteenth International Conference on Machine Learning*, pp. 106-113, Williamstown, MA, 2001. Morgan Kaufmann.

[4] W. Hoeffding. Probability inequalities for sums of bounded random variables. *Journal of the American Statistical Association*, 58:13–30, 1963.

[5] C. Meek, B. Thiesson, and D. Heckerman. The learning-curve method applied to clustering. Technical Report MSR-TR-01-34, Microsoft Research, Redmond, WA, 2000.

[6] B. Thiesson, C. Meek, and D. Heckerman. Accelerating EM for large databases. Technical Report MSR-TR-99-31, Microsoft Research, Redmond, WA, 2001.

[7] A. Wolman, G. Voelker, N. Sharma, N. Cardwell, M. Brown, T. Landray, D. Pinnel, A. Karlin, and H. Levy. Organization-based analysis of Web-object sharing and caching. In *Proceedings of the Second USENIX Conference on Internet Technologies and Systems*, pp. 25–36, Boulder, CO, 1999.

[8] T. Zhang, R. Ramakrishnan, and M. Livny. BIRCH: An efficient data clustering method for very large databases. In *Proceedings of the 1996 ACM SIGMOD International Conference on Management of Data*, pp. 103–114, Montréal, Canada, 1996. ACM Press.
